# AN ELECTRONIC PHOTORECEPTOR

# SENSITIVE TO SMALL CHANGES IN INTENSITY

T. Delbrück and C. A. Mead
256-80 Computer Science
California Institute of Technology
Pasadena, CA 91125

## ABSTRACT

We describe an electronic photoreceptor circuit that is sensitive to small changes in incident light intensity. The sensitivity to *changes* in the intensity is achieved by feeding back to the input a filtered version of the output. The feedback loop includes a hysteretic element. The circuit behaves in a manner reminiscent of the gain control properties and temporal responses of a variety of retinal cells, particularly retinal bipolar cells. We compare the thresholds for detection of intensity increments by a human and by the circuit. Both obey Weber's law and for both the temporal contrast sensitivities are nearly identical.

We previously described an electronic photoreceptor that outputs a voltage that is logarithmic in the light intensity (Mead, 1985). This report describes an extension of this circuit which was based on a suggestion by Frank Werblin that biological retinas may achieve greater sensitivity to *changes* in the illumination by feeding back a filtered version of the output.

## OPERATION OF THE CIRCUIT

The circuit (Figure 1) consists of a phototransistor $(P)$, exponential feedback to $P$ $(Q_1, Q_2,$ and $Q_3)$, a transconductance amplifier $(A)$, and the hysteretic element $(Q_4$ and $Q_5)$. In general terms the operation of the circuit consists of two stages of amplification with hysteresis in the feedback loop. The light falls on the parasitic bipolar transistor $P$. (The rest of the circuit is shielded by metal.) $P$'s collector is the substrate and the base is an isolated well. $P$ and $Q_1$ form the first stage of amplification. The light produces a base current $I_B$ for $P$. The emitter current $I_E$ is $\beta I_B$, neglecting collector resistance for now. $\beta$ is typically a few hundred. The feedback current $I_{Q_1}$ is set by the gate voltage on $Q_1/Q_2$, which is set by the current through $Q_3$, which is set by the feedback voltage $V_{fb}$. In equilibrium $V_{fb}$ will be such that $I_{Q_1} = I_E$ and some voltage $V_P$ will be the output of the first stage. The

negative feedback through the transconductance amplifier $A$ will make $V_P \approx V_{fb}$. This voltage is logarithmic in the light intensity, since in subthreshold operation the currents through $Q_2$ and $Q_3$ are exponential in their gate to source voltages. The DC output of the circuit will be $V_{out} \approx V_{fb} = V_{dd} - (2kT/q)\log I_E$, neglecting the back-gate effect for $Q_2$. Figure 5a (DC output) shows that the assumption of subthreshold operation is valid over about 4 orders of magnitude.

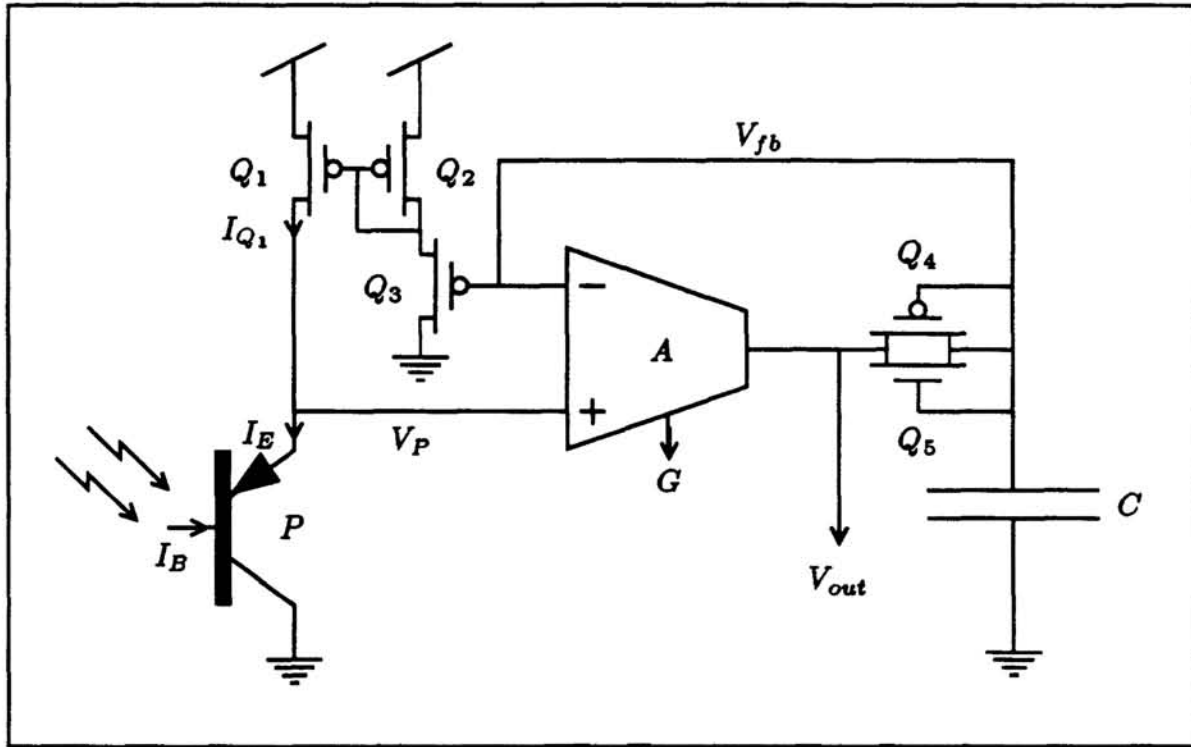

**Figure 1.** The photoreceptor circuit.

Now what happens when the intensity increases a bit? Figure 2a shows the current through $P$ and $Q_1$ as a function of the voltage $V_P$. Both $P$ and $Q_1$ act like current sources in parallel with a resistance, where the value of the current is set, respectively, by the light intensity and by the feedback voltage $V_{fb}$. When the intensity increases a bit the immediate result is that the curve labeled $I_E$ in Figure 2a will shift upwards a little to the curve labeled $I'_E$. But $I_{Q_1}$ won't change right away − it is set by the delayed feedback. The effect on $V_P$ will be that $V_P$ will drop by the amount of the shift in the intersection of the curves in Figure 2a, to $V'_P$. Because interesting gain control properties arise here we will analyze this before going on with the rest of the circuit.

In Figure 2b, we model $P$ and $Q_1$ as current sources with associated drain/collector resistances. Now,

$$\delta V_P = \delta I_E (r_P \| r_{Q_1}) = \delta I_E \frac{r_P r_{Q_1}}{r_P + r_{Q_1}}$$

$r_P$ and $r_{Q_1}$ physically arise from the familiar Early effect, a variation of depletion region thickness causing a variation in the channel length or base thickness. It is a reasonable approximation to model the drain or collector resistance due to the Early effect as $r = V_e/I$, where $V_e$ is the Early voltage and is typically tens of volts, and $I$ is the value of the current source.

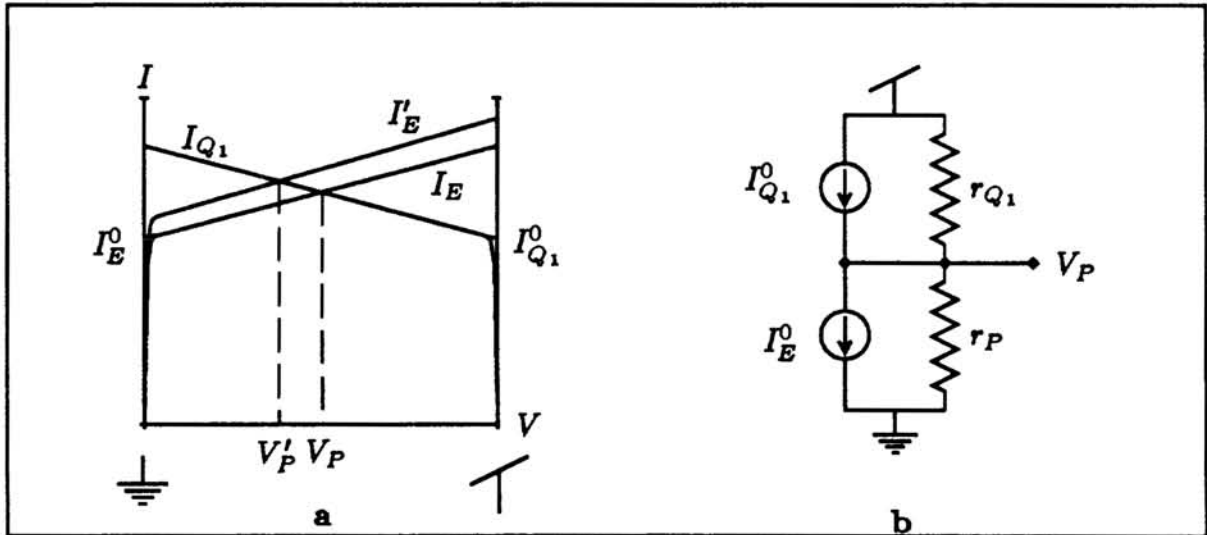

**Figure 2.** The first stage of amplification. a: The curves show the current through the phototransistor $P$ and feedback transistor $Q_1$ as a function of the voltage $V_P$. Since $I_P = I_{Q_1}$ the intersection gives the voltage $V_P$. b: An equivalent circuit model for these transistors in the linear region. The Early effect leads to drain/collector resistances inversely proportional to the value of the current source.

Substituting this approximation for $r_P$ and $r_{Q_1}$ into the above expression for $\delta V_P$ and letting $\delta I_E = \beta \delta I_B$, we obtain

$$\delta V = \frac{\delta I_B}{I_B}(V_{e,P}\|V_{e,Q_1})$$

where $V_{e,P}$ and $V_{e,Q_1}$ are the Early voltages associated with the phototransistor and the feedback transistor $Q_1$, respectively. In other words, the change in $V_P$ is just proportional to the "contrast" $\delta I_B/I_B$. Figure 4a shows test results which support this model.

A detector which encodes the intensity logarithmically (so that the output $V$ in response to an input $I$ is $V = \log I$) would also give $\delta V = \delta I/I$. Our in our circuit the gain control properties for transients arise from an unrelated property of the conductances of the sensor and the feedback element. Comparing the gains for DC and for transients in our circuit and using the expression for the DC output given earlier, we find that the ratio of the gains is

$$\frac{\text{transient gain}}{\text{DC gain}} \approx \frac{V_{e,P}\|V_{e,Q_1}}{2kT/q} \approx 200$$

assuming $V_{e,P} \| V_{e,Q_1} = 10\text{V}$ and $kT/q = 25\text{mV}$.

Finally, let us consider the operation of the rest of the circuit. The second stage of amplification is done by the transconductance amplifier $A$. $A$ produces a current which is proportional to the tanh of the difference between the two inputs, $I = G \tanh(\frac{V_P - V_{fb}}{2kT/q})$. When the output of the amplifier is taken as a voltage, the voltage gain is typically a few hundred. Following the transconductance amplifier there is a pair of diode connected transistors, $Q_4$ and $Q_5$ (Figure 3a), which we call the hysteretic element. This pair of transistors has an I-V characteristic which is similar to that of Figure 3. The hysteretic element conducts very little until the voltage across it becomes substantial. Thus the transconductance amplifier works in a dual voltage-output/current-output mode. Small changes in the output voltage result in little change in the feedback voltage. Larger changes in the output voltage cause current to flow through the hysteretic element, completing the feedback loop. This represents a form of memory, or hysteresis, for the past state of the output and a sensitivity to small changes in the input around the past history of the input.

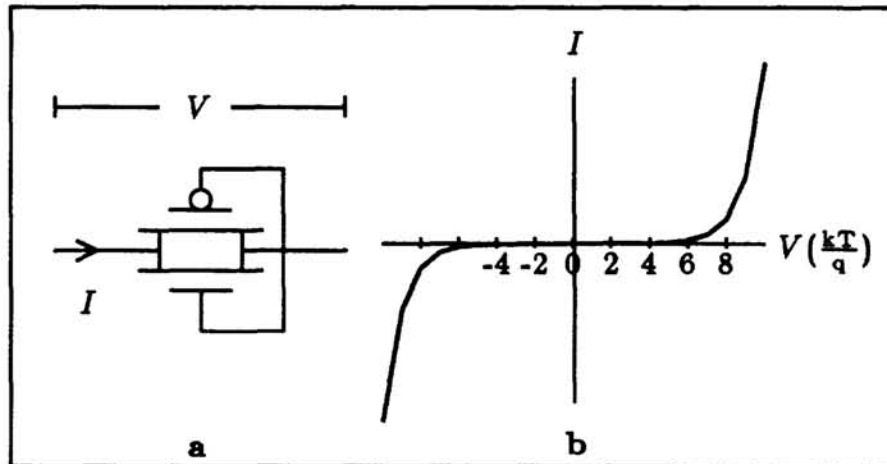

Figure 3. a: The hysteretic element. b: $I - V$ characteristic.

## COMPARISON OF CIRCUIT AND RETINAL CELLS

We felt that since the circuit was motivated by biology it might be interesting to compare the operational characteristics of the circuit and of retinal cells. Since the circuit has no spatial extent it cannot model any of the spatially mediated effects (such as center–surround) seen in retinas. Nonetheless we had hoped to capture some of the temporal effects seen in retinal cells. In Figure 4 we compare the responses of the circuit and responses of a retinal bipolar cell to diffuse flashes of light. The circuit has response characteristics closest to those of retinal bipolar cells.

The circuit's gain *control* properties are very similar to those of bipolar cells. Figure 5 shows that both the circuit and bipolar cells tend to control their gain so that they maintain a constant output amplitude for a given change in the *log* of the intensity.

The response characteristics of the circuit differ from those of bipolar cells in the following ways. First, the gain of the circuit for transients is much larger than that of the bipolar cell, as can be seen in Figure 5, with concomitantly much smaller dynamic range. The dynamic range of the bipolar cell is about $1.5 - 2$ log units around the steady intensity, while for the circuit the dynamic range is only about 0.1 log unit.

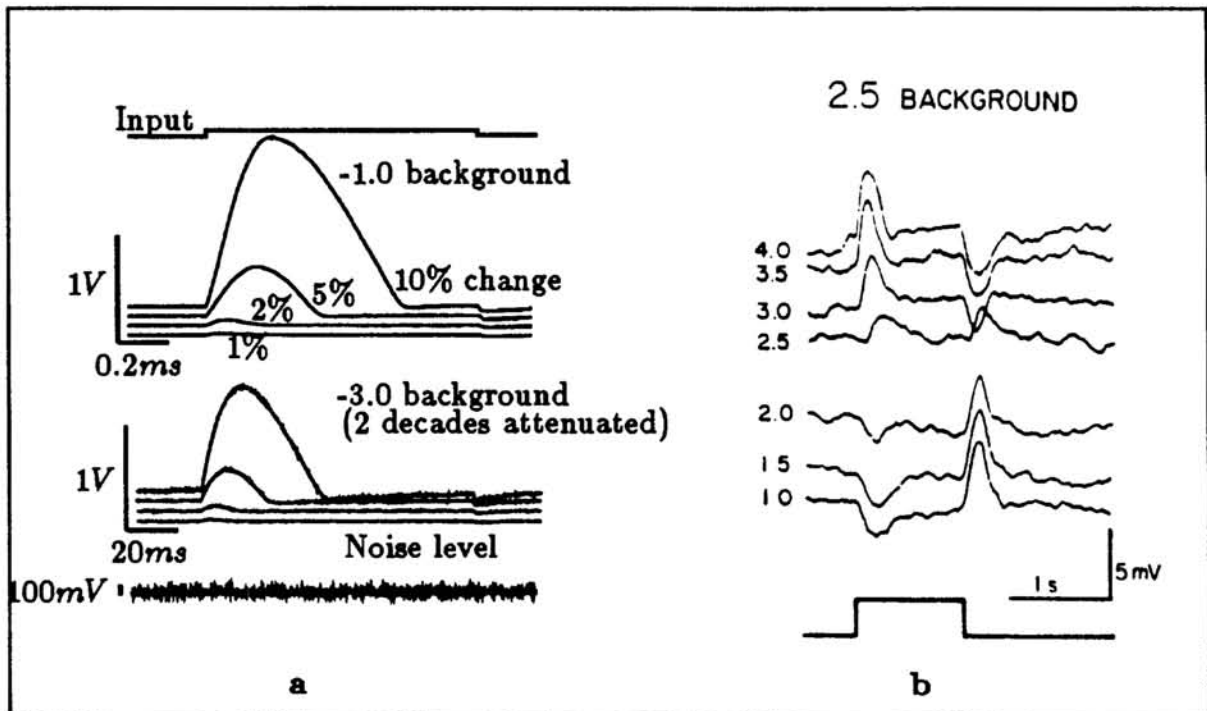

**Figure 4.** The responses of the circuit compared with responses of a retinal bipolar cell. **a**: The output of the circuit in response to changes in the intensity. The background levels refer to the same scale as shown in Figure 5. The bottom curve shows the noise level; from this one can see that a detection criterion of signal/noise ratio equals 2 is satisfied for increments of $1-2\%$, in agreement with Figure 6. Note that a 2 decade attenuation hardly changes the response amplitude but the time constant increases by a factor of a hundred. **b**: The response of a bipolar cell (from Werblin, 1974). The numbers next to the responses are the log of the intensity of the flash substituted for the intial value of the intensity. Note the bidirectionality of the response compared to the circuit.

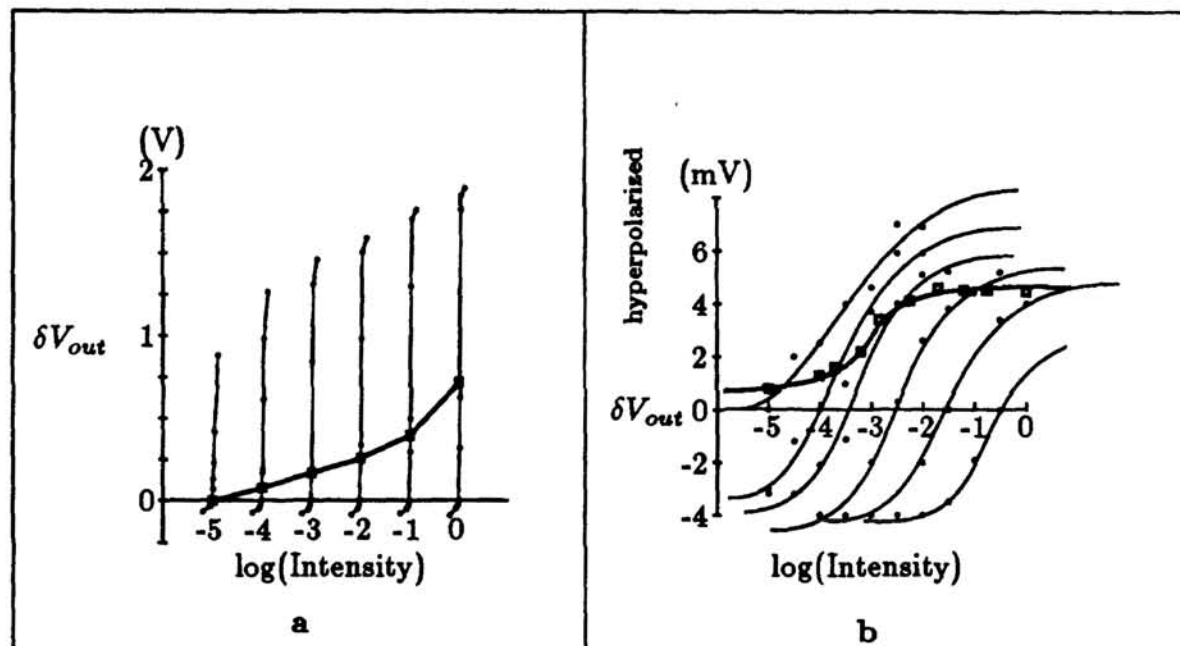

**Figure 5.** The operating curves of the circuit (a) compared with retinal bipolar cells (b) (adapted from Werblin, 1974). The curves show the height of the peak of the response to flashes substituted for the initial intensity. The initial intensity is given by the intersection of the curves with the abscissa. Note the difference of the gain and the dynamic range. The squares show the DC responses. The slope of the DC response for the circuit is less than expected, probably because there is a leakage current through the hysteretic element.

Second, the response of bipolar cells is symmetrical for increases and decreases in the intensity. This can probably be traced to the symmetrical responses of the cones from which they receive direct input. The circuit, on the other hand, only responds strongly to increases in the light intensity. The response in our circuit only becomes symmetrical for output voltage swings comparable to $kT/q$, probably because the limiting process is recombination in the base of the phototransistor.

Third, the control of time constants is dramatically different. In Figure 4a the top set of responses is on a time scale 100 times expanded relative to the bottom scale. The circuit's time constant, in other words, is roughly inversely proportional to the light intensity. This is not the case for bipolar cells. Although we do not show it here, the time constant of the responses of bipolar cells hardly varies with light intensity over at least 4 orders of magnitude (Werblin, 1974).

The circuit's action differs much more from that of photoreceptors, amacrine, or ganglion cells. Cones show a much larger sustained response relative to their transient response. Amacrine and ganglion cells spike; our circuit does not. And the circuit differs from on/off amacrine and ganglion cells in the asymmetry of its response to increases and decreases in light intensity.

## EYE *vs.* CHIP

We compared the sensitivity to small changes in the light intensity for one of us and for the circuit in order to get an idea of the performance of the circuit relative to a subjective scale. The thresholds for detection of intensity increments are shown in Figure 6.

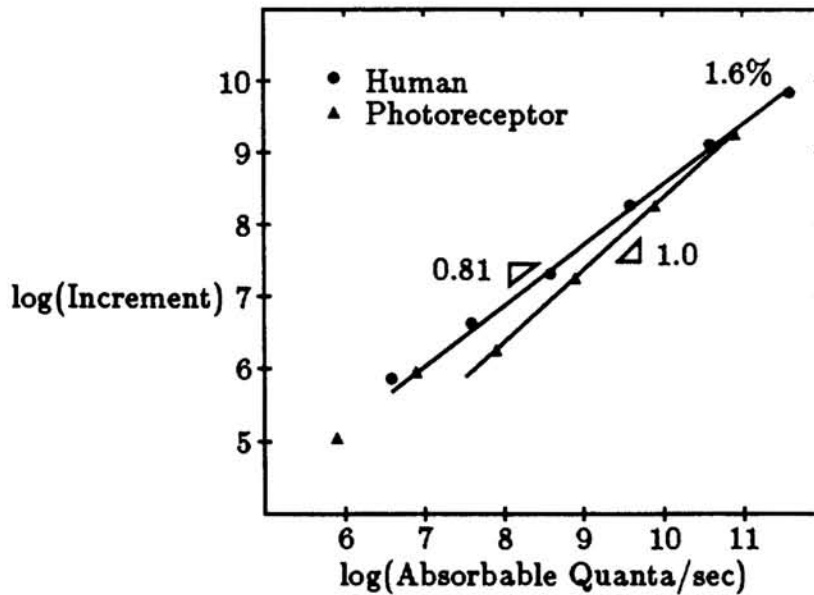

**Figure 6.** Thresholds for detection of flicker.

The subject (TD) sat in a darkened room foveating a flickering yellow (583nm) LED at a distance of 75cm. The LED subtended 22' of arc and the frequency of flicker was 5 Hz. Threshold determination was made by a series of trials, indicated by a buzzer, in which the computer either caused the LED to flicker or not to flicker. The percentage of flicker was started at some large amount for which determination was unambiguous. For each trial the subject pressed a button if he thought he saw the LED flickering. A incorrect response would cause the percentage of flicker for the next trial to be increased. A correct response would cause the percentage of flicker to be decreased with a probability of 0.44. The result after a hundred trials would be a curve of percentage flicker vs. trial number which started high and then leveled off around the 75% correct level, taken to be the threshold (Levitt, 1971).

The threshold for the circuit was determined by shining the same LED onto the chip directly. The threshold was defined as the smallest amount of flicker that would result in a signal-to-noise ratio of 2 for the output. The two sets of thresholds were scaled relative to each other using a Tektronics photometer and were both scaled to read in terms of absorbable quanta, defined for the human as the number of photons hitting the cornea and for the circuit as the number of quanta that hit the area of the phototransistor. There is a bias here favoring the circuit, since the other parts

of the circuit have not been included in this area. Including the rest of the circuit would raise the thresholds for the circuit by a factor of about 3.

The results show that both the circuit and the human approximately obey Weber's law ($\delta I/I$ at threshold is a constant), and the sensitivities are nearly the same. The highest sensitivity for the circuit, measured at an incident intensity of $660\mu W/cm^2$, was 1.2%. This is about half the intensity in a brightly lit office.

# APPLICATIONS

We are trying to develop these sensors for use in neurophysiological optical dye recording. These experiments require a sensor capable of recording changes in the incident intensity of about 1 part in $10^{3-4}$ (Grinvald, 1985). The current technique is to use integrated arrays of photodiodes each with a dedicated rack-mounted low noise amplifier. We will try to replace this arrangement with arrays of receptors of the type discussed in this paper. Currently we are 1 to 2 orders of magnitude short of the required sensitivity, but we hope to improve the performance by using hybrid bipolar/FET technology.

# CONCLUSION

This circuit represents an example of an idea from biology directly and simply synthesized in silicon. The resulting circuit incorporated not only the intended idea, sensitivity to changes in illumination, but also gave rise to an unexpected gain control mechanism unrelated to exponential feedback. The circuit differs in several ways from its possible biological analogy but remains an interesting and potentially useful device.

### Acknowledgements

This work was supported by the System Development Foundation and the Office of Naval Research. Chips were fabricated through the MOSIS foundry.

We thank Frank Werblin for helpful comments and Mary Ann Maher for editorial assistance.

### References

A. Grinvald. (1985) Real time optical mapping of neuronal activity: from single growth cones to the intact mammalian brain. *Ann. Rev. Neurosci.* **8**:263–305.

H. Levitt. (1971) Transformed up-down methods in psychoacoustics. *J. Acoust. Soc. Am.* **49**:467-477.

C. Mead. (1985) A Sensitive Electronic Photoreceptor. In *1985 Chapel Hill Conference on VLSI.* 463–471.

F. Werblin. (1974) Control of Retinal Sensitivity II: Lateral Interactions at the Outer Plexiform Layer. *J. Physiology.* **63**:62–87.